# Learning Bayesian Networks with Thousands of Variables

**Mauro Scanagatta**
IDSIA*, SUPSI†, USI‡
Lugano, Switzerland
mauro@idsia.ch

**Cassio P. de Campos**
Queen's University Belfast
Northern Ireland, UK
c.decampos@qub.ac.uk

**Giorgio Corani**
IDSIA*, SUPSI†, USI‡
Lugano, Switzerland
giorgio@idsia.ch

**Marco Zaffalon**
IDSIA*
Lugano, Switzerland
zaffalon@idsia.ch

## Abstract

We present a method for learning Bayesian networks from data sets containing thousands of variables without the need for structure constraints. Our approach is made of two parts. The first is a novel algorithm that effectively explores the space of possible parent sets of a node. It guides the exploration towards the most promising parent sets on the basis of an approximated score function that is computed in constant time. The second part is an improvement of an existing ordering-based algorithm for structure optimization. The new algorithm provably achieves a higher score compared to its original formulation. Our novel approach consistently outperforms the state of the art on very large data sets.

## 1 Introduction

Learning the structure of a Bayesian network from data is NP-hard [2]. We focus on score-based learning, namely finding the structure which maximizes a score that depends on the data [9]. Several exact algorithms have been developed based on dynamic programming [12, 17], branch and bound [7], linear and integer programming [4, 10], shortest-path heuristic [19, 20].

Usually structural learning is accomplished in two steps: *parent set identification* and *structure optimization*. Parent set identification produces a list of suitable candidate parent sets for each variable. *Structure optimization* assigns a parent set to each node, maximizing the score of the resulting structure without introducing cycles.

The problem of *parent set identification* is unlikely to admit a polynomial-time algorithm with a good quality guarantee [11]. This motivates the development of effective search heuristics. Usually however one decides the maximum in-degree (number of parents per node) $k$ and then simply computes the score of all parent sets. At that point one performs structural optimization. An exception is the greedy search of the K2 algorithm [3], which has however been superseded by the more modern approaches mentioned above.

A higher in-degree implies a larger search space and allows achieving a higher score; however it also requires higher computational time. When choosing the in-degree the user makes a trade-off between these two objectives. However when the number of variables is large, the in-degree is

generally set to a small value, to allow the optimization to be feasible. The largest data set analyzed in [1] with the *Gobnilp*[1] software contains 413 variables; it is analyzed setting $k = 2$. In [5] Gobnilp is used for structural learning with 1614 variables, setting $k = 2$. These are among the largest examples of score-based structural learning in the literature.

In this paper we propose an algorithm that performs approximated structure learning with *thousands* of variables *without* constraints on the in-degree. It is constituted by a novel approach for parent set identification and a novel approach for structure optimization.

As for *parent set identification* we propose an anytime algorithm that effectively explores the space of possible parent sets. It guides the exploration towards the most promising parent sets, exploiting an approximated score function that is computed in constant time. As for *structure optimization*, we extend the ordering-based algorithm of [18], which provides an effective approach for model selection with reduced computational cost. Our algorithm is guaranteed to find a solution better than or equal to that of [18].

We test our approach on data sets containing up to ten thousand variables. As a performance indicator we consider the score of the network found. Our parent set identification approach outperforms consistently the usual approach of setting the maximum in-degree and then computing the score of all parent sets. Our structure optimization approach outperforms Gobnilp when learning with more than 500 nodes. All the software and data sets used in the experiments are available online. [2].

## 2    Structure Learning of Bayesian Networks

Consider the problem of learning the structure of a Bayesian Network from a complete data set of $N$ instances $\mathcal{D} = \{D_1, ..., D_N\}$. The set of $n$ categorical random variables is $\mathcal{X} = \{X_1, ..., X_n\}$. The goal is to find the best DAG $\mathcal{G} = (\mathcal{V}, \mathcal{E})$, where $\mathcal{V}$ is the collection of nodes and $\mathcal{E}$ is the collection of arcs. $\mathcal{E}$ can be defined as the set of parents $\Pi_1, ..., \Pi_n$ of each variable. Different scores can be used to assess the fit of a DAG. We adopt the BIC, which asymptotically approximates the posterior probability of the DAG. The BIC score is *decomposable*, namely it is constituted by the sum of the scores of the individual variables:

$$\text{BIC}(\mathcal{G}) =$$
$$= \sum_{i=1}^{n} \text{BIC}(X_i, \Pi_i) = \sum_{i=1}^{n} \sum_{\pi \in |\Pi_i|} \sum_{x \in |X_i|} N_{x,\pi} \log \hat{\theta}_{x|\pi} - \frac{\log N}{2} (|X_i| - 1)(|\Pi_i|) \,,$$

where $\hat{\theta}_{x|\pi}$ is the maximum likelihood estimate of the conditional probability $P(X_i = x|\Pi_i = \pi)$, and $N_{x,\pi}$ represents the number of times $(X = x \wedge \Pi_i = \pi)$ appears in the data set, and $|\cdot|$ indicates the size of the Cartesian product space of the variables given as arguments (instead of the number of variables) such that $|X_i|$ is the number of states of $X_i$ and $|\varnothing| = 1$.

Exploiting decomposability, we first identify independently for each variable a list of candidate parent sets (parent set identification). Then by structure optimization we select for each node the parent set that yields the highest score without introducing cycles.

## 3    Parent set identification

For *parent set identification* usually one explores all the possible parent sets, whose number however increases as $O(n^k)$, where $k$ denotes the maximum in-degree. Pruning rules [7] do not considerably reduce the size of this space.

Usually the parent sets are explored in sequential order: first all the parent size of size one, then all the parent sets of size two, and so on, up to size $k$. We refer to this approach as *sequential ordering*. If the solver adopted for structural optimization is exact, this strategy allows to find the globally optimum graph *given* the chosen value of $k$. In order to deal with a large number of variables it is however necessary setting a low in-degree $k$. For instance [1] adopts $k$=2 when dealing with the largest data set (diabetes), which contains 413 variables. In [5] Gobnilp is used for structural learning with 1614 variables, again setting $k = 2$. A higher value of $k$ would make the structural

learning not feasible. Yet a low $k$ implies dropping all the parent sets with size larger than $k$. Some of them possibly have a high score.

In [18] it is proposed to adopt the subset $\Pi_{\text{corr}}$ of the most correlated variables with the children variable. Then [18] consider only parent sets which are subsets of $\Pi_{\text{corr}}$. However this approach is not commonly adopted, possibly because it requires specifying the size of $\Pi_{\text{corr}}$. Indeed [18] acknowledges the need for further innovative approaches in order to effectively explore the space of the parent sets.

We propose two anytime algorithms to address this problem. The first is the simplest; we call it *greedy selection*. It starts by exploring all the parent sets of size one and adding them to a list. Then it repeats the following until time is expired: pops the best scoring parent set $\Pi$ from the list, explores all the supersets obtained by adding one variable to $\Pi$, and adds them to the list. Note that in general the parent sets chosen at two adjoining step are not related to each other. The second approach (*independence selection*) adopts a more sophisticated strategy, as explained in the following.

### 3.1 Parent set identification by independence selection

Independence selection uses an approximation of the actual BIC score of a parent set $\Pi$, which we denote as $\text{BIC}^*$, to guide the exploration of the space of the parent sets. The $\text{BIC}^*$ of a parent set constituted by the union of two non-empty parent sets $\Pi_1$ and $\Pi_2$ is defined as follows:

$$\text{BIC}^*(X, \Pi_1, \Pi_2) = \text{BIC}(X, \Pi_1) + \text{BIC}(X, \Pi_2) + \text{inter}(X, \Pi_1, \Pi_2), \tag{1}$$

with $\Pi_1 \cup \Pi_2 = \Pi$ and $\text{inter}(X, \Pi_1, \Pi_2) = \frac{\log N}{2}(|X|-1)(|\Pi_1|+|\Pi_2|-|\Pi_1||\Pi_2|-1)-\text{BIC}(X, \varnothing)$. If we already know $\text{BIC}(X, \Pi_1)$ and $\text{BIC}(X, \Pi_2)$ from previous calculations (and we know $\text{BIC}(X, \varnothing)$), then $\text{BIC}^*$ can be computed in constant time (with respect to data accesses). We thus exploit $\text{BIC}^*$ to quickly estimate the score of a large number of candidate parent sets and to decide the order to explore them.

We provide a bound for the difference between $\text{BIC}^*(X, \Pi_1, \Pi_2)$ and $\text{BIC}(X, \Pi_1 \cup \Pi_2)$. To this end, we denote by ii the *Interaction Information* [14]: $\text{ii}(X; Y; Z) = \text{I}(X; Y|Z) - \text{I}(X; Y)$, namely the difference between the mutual information of $X$ and $Y$ conditional on $Z$ and the unconditional mutual information of $X$ and $Y$.

**Theorem 1.** *Let $X$ be a node of $\mathcal{G}$ and $\Pi = \Pi_1 \cup \Pi_2$ be a parent set for $X$ with $\Pi_1 \cap \Pi_2 = \varnothing$ and $\Pi_1, \Pi_2$ non-empty. Then $\text{BIC}(X, \Pi) = \text{BIC}^*(X, \Pi_1, \Pi_2) + N \cdot \text{ii}(\Pi_1; \Pi_2; X)$, where ii is the Interaction Information estimated from data.*

*Proof.* $\text{BIC}(X, \Pi_1 \cup \Pi_2) - \text{BIC}^*(X, \Pi_1, \Pi_2) =$

$$\text{BIC}(X, \Pi_1 \cup \Pi_2) - \text{BIC}(X, \Pi_1) - \text{BIC}(X, \Pi_2) - \text{inter}(X, \Pi_1, \Pi_2) =$$

$$\sum\nolimits_{x, \pi_1, \pi_2} N_{x, \pi_1, \pi_2} \left[ \log \hat{\theta}_{x|\pi_1, \pi_2} - \log(\hat{\theta}_{x|\pi_1} \hat{\theta}_{x|\pi_2}) \right] + \sum\nolimits_x N_x \log \hat{\theta}_x =$$

$$\sum\nolimits_{x, \pi_1, \pi_2} N_{x, \pi_1, \pi_2} \left[ \log \hat{\theta}_{x|\pi_1, \pi_2} - \log \left( \frac{\hat{\theta}_{x|\pi_1} \hat{\theta}_{x|\pi_2}}{\hat{\theta}_x} \right) \right] =$$

$$\sum\nolimits_{x, \pi_1, \pi_2} N_{x, \pi_1, \pi_2} \log \left( \frac{\hat{\theta}_{x|\pi_1, \pi_2} \hat{\theta}_x}{\hat{\theta}_{x|\pi_1} \hat{\theta}_{x|\pi_2}} \right) = \sum\nolimits_{x, \pi_1, \pi_2} N \cdot \hat{\theta}_{x, \pi_1, \pi_2} \log \left( \frac{\hat{\theta}_{\pi_1, \pi_2|x} \hat{\theta}_{\pi_1} \hat{\theta}_{\pi_2}}{\hat{\theta}_{\pi_1|x} \hat{\theta}_{\pi_2|x} \hat{\theta}_{\pi_1, \pi_2}} \right) =$$

$$N \left( \sum\nolimits_{x, \pi_1, \pi_2} \hat{\theta}_{x, \pi_1, \pi_2} \log \left( \frac{\hat{\theta}_{\pi_1, \pi_2|x}}{\hat{\theta}_{\pi_1|x} \hat{\theta}_{\pi_2|x}} \right) - \sum\nolimits_{\pi_1, \pi_2} \hat{\theta}_{\pi_1, \pi_2} \log \left( \frac{\hat{\theta}_{\pi_1, \pi_2}}{\hat{\theta}_{\pi_1} \hat{\theta}_{\pi_2}} \right) \right) =$$

$$N \cdot (\text{I}(\Pi_1; \Pi_2|X) - \text{I}(\Pi_1; \Pi_2)) = N \cdot \text{ii}(\Pi_1; \Pi_2; X),$$

where $\text{I}(\cdot)$ denotes the (conditional) mutual information estimated from data. $\qquad \square$

**Corollary 1.** *Let $X$ be a node of $\mathcal{G}$, and $\Pi = \Pi_1 \cup \Pi_2$ be a parent set of $X$ such that $\Pi_1 \cap \Pi_2 = \varnothing$ and $\Pi_1, \Pi_2$ non-empty. Then*

$$|\text{BIC}(X, \Pi) - \text{BIC}^*(X, \Pi_1, \Pi_2)| \leq N \min\{\text{H}(X), \text{H}(\Pi_1), \text{H}(\Pi_2)\}.$$

*Proof.* Theorem 1 states that $\text{BIC}(X, \Pi) = \text{BIC}^*(X, \Pi_1, \Pi_2) + N \cdot \text{ii}(\Pi_1; \Pi_2; X)$. We now devise bounds for interaction information, recalling that mutual information and conditional mutual information are always non-negative and achieve their maximum value at the smallest entropy H of their

argument: $-\mathrm{H}(\Pi_2) \leq -\mathrm{I}(\Pi_1; \Pi_2) \leq \mathrm{ii}(\Pi_1; \Pi_2; X) \leq \mathrm{I}(\Pi_1; \Pi_2 | X) \leq \mathrm{H}(\Pi_2)$. The theorem is proven by simply permuting the values $\Pi_1; \Pi_2; X$ in the ii of such equation. Since

$$\mathrm{ii}(\Pi_1; \Pi_2; X) = \mathrm{I}(\Pi_1; \Pi_2 | X) - \mathrm{I}(\Pi_1; \Pi_2) = \mathrm{I}(X; \Pi_1 | \Pi_2) - \mathrm{I}(X; \Pi_1) = \mathrm{I}(\Pi_2; X | \Pi_1) - \mathrm{I}(\Pi_2; X),$$

the bounds for ii are valid. $\qquad\square$

We know that $0 \leq \mathrm{H}(\Pi) \leq \log(|\Pi|)$ for any set of nodes $\Pi$, hence the result of Corollary 1 could be further manipulated to achieve a bound for the difference between BIC and BIC$^*$ of at most $N \log(\min\{|X|, |\Pi_1|, |\Pi_2|\})$. However, Corollary 1 is stronger and can still be computed efficiently as follows. When computing $\mathrm{BIC}^*(X, \Pi_1, \Pi_2)$, we assumed that $\mathrm{BIC}(X, \Pi_1)$ and $\mathrm{BIC}(X, \Pi_2)$ had been precomputed. As such, we can also have precomputed the values $\mathrm{H}(\Pi_1)$ and $\mathrm{H}(\Pi_2)$ at the same time as the BIC scores were computed, without any significant increase of complexity (when computing $\mathrm{BIC}(X, \Pi)$ for a given $\Pi$, just use the same loop over the data to compute $\mathrm{H}(\Pi)$).

**Corollary 2.** *Let $X$ be a node of $\mathcal{G}$, and $\Pi = \Pi_1 \cup \Pi_2$ be a parent set for that node with $\Pi_1 \cap \Pi_2 = \varnothing$ and $\Pi_1, \Pi_2$ non-empty. If $\Pi_1 \perp \Pi_2$, then $\mathrm{BIC}(X, \Pi_1 \cup \Pi_2) \geq \mathrm{BIC}^*(X, \Pi_1 \cup \Pi_2)$. If $\Pi_1 \perp \Pi_2 \, | X$, then $\mathrm{BIC}(X, \Pi_1 \cup \Pi_2) \leq \mathrm{BIC}^*(X, \Pi_1 \cup \Pi_2)$. If the interaction information $\mathrm{ii}(\Pi_1; \Pi_2; X) = 0$, then $\mathrm{BIC}(X, \Pi_1 \cup \Pi_2) = \mathrm{BIC}^*(X, \Pi_1, \Pi_2)$.*

*Proof.* It follows from Theorem 1 considering that mutual information $\mathrm{I}(\Pi_1, \Pi_2) = 0$ if $\Pi_1$ and $\Pi_2$ are independent, while $\mathrm{I}(\Pi_1, \Pi_2 | X) = 0$ if $\Pi_1$ and $\Pi_2$ are conditionally independent. $\qquad\square$

We now devise a novel pruning strategy for BIC based on the bounds of Corollaries 1 and 2.

**Theorem 2.** *Let $X$ be a node of $\mathcal{G}$, and $\Pi = \Pi_1 \cup \Pi_2$ be a parent set for that node with $\Pi_1 \cap \Pi_2 = \varnothing$ and $\Pi_1, \Pi_2$ non-empty. Let $\Pi' \supset \Pi$. If $\mathrm{BIC}^*(X, \Pi_1, \Pi_2) + \frac{\log N}{2}(|X| - 1)|\Pi'| > N \min\{\mathrm{H}(X), \mathrm{H}(\Pi_1), \mathrm{H}(\Pi_2)\}$, then $\Pi'$ and its supersets are not optimal and can be ignored.*

*Proof.* $\mathrm{BIC}^*(X, \Pi_1, \Pi_2) - N \min\{\mathrm{H}(X), \mathrm{H}(\Pi_1), \mathrm{H}(\Pi_2)\} + \frac{\log N}{2}(|X| - 1)|\Pi'| > 0$ implies $\mathrm{BIC}(\Pi) + \frac{\log N}{2}(|X| - 1)|\Pi'| > 0$, and Theorem 4 of [6] prunes $\Pi'$ and all its supersets. $\qquad\square$

Thus we can efficiently check whether large parts of the search space can be discarded based on these results. We note that Corollary 1 and hence Theorem 2 are very generic in the choice of $\Pi_1$ and $\Pi_2$, even though usually one of them is taken as a singleton.

### 3.2 Independence selection algorithm

We now describe the algorithm that exploits the BIC$^*$ score in order to effectively explore the space of the parent sets. It uses two lists: (1) *open*: a list for the parent sets to be explored, ordered by their BIC$^*$ score; (2) *closed*: a list of already explored parent sets, along with their actual BIC score.

The algorithm starts with the BIC of the empty set computed. First it explores all the parent sets of size one and saves their BIC score in the *closed* list. Then it adds to the *open* list every parent set of size two, computing their BIC$^*$ scores in constant time on the basis of the scores available from the *closed* list. It then proceeds as follows until all elements in *open* have been processed, or the time is expired. It extracts from *open* the parent set $\Pi$ with the best BIC$^*$ score; it computes its BIC score and adds it to the *closed* list. It then looks for all the possible expansions of $\Pi$ obtained by adding a single variable $Y$, such that $\Pi \cup Y$ is not present in *open* or *closed*. It adds them to *open* with their $\mathrm{BIC}^*(X, \Pi, Y)$ scores. Eventually it also considers all the explored subsets of $\Pi$. It safely [7] prunes $\Pi$ if any of its subsets yields a higher BIC score than $\Pi$. The algorithm returns the content of the *closed* list, pruned and ordered by the BIC score. Such list becomes the content of the so-called *cache of scores* for $X$. The procedure is repeated for every variable and can be easily parallelized.

Figure 1 compares sequential ordering and independence selection. It shows that independence selection is more effective than sequential ordering because it biases the search towards the highest-scoring parent sets.

## 4 Structure optimization

The goal of structure optimization is to choose the overall highest scoring parent sets (measured by the sum of the local scores) without introducing directed cycles in the graph. We start from the approach proposed in [18] (which we call *ordering-based search* or OBS), which exploits the fact

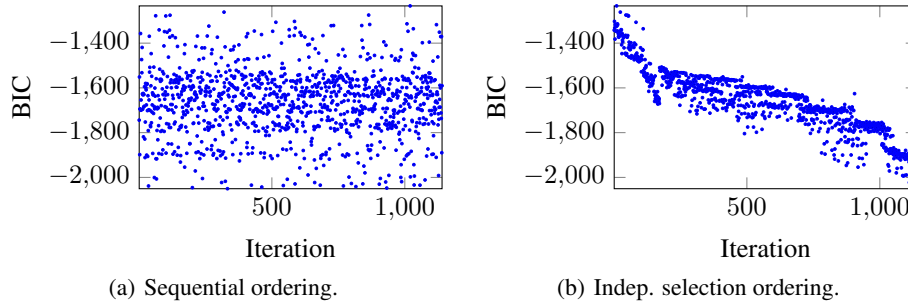

(a) Sequential ordering.          (b) Indep. selection ordering.

Figure 1: Exploration of the parent sets space for a given variable performed by sequential ordering and independence selection. Each point refers to a distinct parent set.

that the optimal network can be found in time $O(Ck)$, where $C = \sum_{i=1}^{n} c_i$ and $c_i$ is the number of elements in the *cache of scores* of $X_i$, if an ordering over the variables is given.[3] $\Theta(k)$ is needed to check whether all the variables in a parent set for $X$ come before $X$ in the ordering (a simple array can be used as data structure for this checking). This implies working on the search space of the possible orderings, which is convenient as it is smaller than the space of network structures. Multiple orderings are sampled and evaluated (different techniques can be used for guiding the sampling). For each sampled total ordering $\prec$ over variables $X_1, \ldots, X_n$, the network is consistent with the order if $\forall X_i : \forall X \in \Pi_i : X \prec X_i$. A network consistent with a given ordering automatically satisfies the acyclicity constraint. This allows us to choose independently the best parent set of each node. Moreover, for a given total ordering $V_1, \ldots, V_n$ of the variables, the algorithm tries to improve the network by a greedy search *swapping* procedure: if there is a pair $V_j, V_{j+1}$ such that the swapped ordering with $V_j$ in place of $V_{j+1}$ (and vice versa) yields better score for the network, then these nodes are swapped and the search continues. One advantage of this swapping over extra random orderings is that searching for it and updating the network (if a good swap is found) only takes time $O((c_j + c_{j+1}) \cdot kn)$ (which can be sped up as $c_j$ only is inspected for parents sets containing $V_{j+1}$, and $c_{j+1}$ is only processed if $V_{j+1}$ has $V_j$ as parent in the current network), while a new sampled ordering would take $O(n + Ck)$ (the swapping approach is usually favourable if $c_i$ is $\Omega(n)$, which is a plausible assumption). We emphasize that the use of $k$ here is sole with the purpose of analyzing the complexity of the methods, since our parent set identification approach does not rely on a fixed value for $k$.

However, the consistency rule of OBS is quite restricting. While it surely refuses all cyclic structures, it also rules out some acyclic ones which could be captured by interpreting the ordering in a slightly different manner. We propose a novel consistency rule for a given ordering which processes the nodes in $V_1, \ldots, V_n$ from $V_n$ to $V_1$ (OBS can do it in any order, as the local parent sets can be chosen independently) and we define the parent set of $V_j$ such that it does not introduce a cycle in the current partial network. This allows back-arcs in the ordering from a node $V_j$ to its successors, as long as this does not introduce a cycle. We call this idea *acyclic selection OBS* (or simply ASOBS). Because we need to check for cycles at each step of constructing the network for a given ordering, at a first glance the algorithm seems to be slower (time complexity of $O(Cn)$ against $O(Ck)$ for OBS; note this difference is only relevant as we intend to work with large values $n$). Surprisingly, we can implement it in the same overall time complexity of $O(Ck)$ as follows.

1. Build and keep a Boolean square matrix $m$ to mark which are the descendants of nodes ($m(X, Y)$ tells whether $Y$ is descendant of $X$). Start it all *false*.
2. For each node $V_j$ in the order, with $j = n, \ldots, 1$:
   (a) Go through the parent sets and pick the best scoring one for which all contained parents are not descendants of $V_j$ (this takes time $O(c_i k)$ if parent sets are kept as lists).
   (b) Build a *todo* list with the descendants of $V_j$ from the matrix representation and associate an empty *todo* list to all ancestors of $V_j$.
   (c) Start the *todo* lists of the parents of $V_j$ with the descendants of $V_j$.
   (d) For each ancestor $X$ of $V_j$ (ancestors will be iteratively visited by following a depth-first graph search procedure using the network built so far; we process a node after

its children with non-empty *todo* lists have been already processed; the search stops when all ancestors are visited):

     i. For each element $Y$ in the *todo* list of $X$, if $m(X, Y)$ is true, then ignore $Y$ and move on; otherwise set $m(X, Y)$ to true and add $Y$ to the *todo* of parents of $X$.

Let us analyze the complexity of the method. Step 2a takes overall time $O(Ck)$ (already considering the outer loop). Step 2b takes overall time $O(n^2)$ (already considering the outer loop). Steps 2c and 2(d)i will be analyzed based on the number of elements on the *todo* lists and the time to process them in an amortized way. Note that the time complexity is directly related to the number of elements that are processed from the *todo* lists (we can simply look to the moment that they leave a list, as their inclusion in the lists will be in equal number). We will now count the number of times we process an element from a *todo* list. This number is overall bounded (over all external loop cycles) by the number of times we can make a cell of matrix $m$ turn from false to true (which is $O(n^2)$) plus the number of times we ignore an element because the matrix cell was already set to true (which is at most $O(n)$ per each $V_j$, as this is the maximum number of descendants of $V_j$ and each of them can fall into this category only once, so again there are $O(n^2)$ times in total). In other words, each element being removed from a *todo* list is either ignored (matrix already set to true) or an entry in the matrix of descendants is changed from false to true, and this can only happen $O(n^2)$ times. Hence the total time complexity is $O(Ck + n^2)$, which is $O(Ck)$ for any $C$ greater than $n^2/k$ (a very plausible scenario, as each local *cache* of a variable usually has more than $n/k$ elements).

Moreover, we have the following interesting properties of this new method.

**Theorem 3.** *For a given ordering $\prec$, the network obtained by ASOBS has score equal than or greater to that obtained by OBS.*

*Proof.* It follows immediately from the fact that the consistency rule of ASOBS generalizes that of OBS, that is, for each node $V_j$ with $j = n, \ldots, 1$, ASOBS allows all parent sets allowed by OBS and also others (containing back-arcs). □

**Theorem 4.** *For a given ordering $\prec$ defined by $V_1, \ldots, V_n$ and a current graph $\mathcal{G}$ consistent with $\prec$, if OBS consistency rule allows the swapping of $V_j, V_{j+1}$ and leads to improving the score of $\mathcal{G}$, then the consistency rule of ASOBS allows the same swapping and achieves the same improvement in score.*

*Proof.* It follows immediately from the fact that the consistency rule of ASOBS generalizes that of OBS, so from a given graph $\mathcal{G}$, if a swapping is possible under OBS rules, then it is also possible under ASOBS rules. □

## 5 Experiments

We compare three different approaches for parent set identification (sequential, greedy selection and independence selection) and three different approaches (Gobnilp, OBS and ASOBS) for structure optimization. This yields nine different approaches for structural learning, obtained by combining all the methods for parent set identification and structure optimization. Note that OBS has been shown in [18] to outperform other greedy-tabu search over structures, such as greedy hill-climbing and optimal-reinsertion-search methods [15].

We allow one minute per variable to each approach for parent set identification. We set the maximum in-degree to $k = 6$, a high value that allows learning even complex structures. Notice that our novel approach does not need a maximum in-degree. We set a maximum in-degree to put our approach and its competitors on the same ground. Once computed the scores of the parent sets we run each solver (Gobnilp, OBS, ASOBS) for 24 hours. For a given data set the computation is performed on the same machine.

The explicit goal of each approach for both parent set identification and structure optimization is to maximize the BIC score. We then measure the BIC score of the Bayesian networks eventually obtained as performance indicator. The difference in the BIC score between two alternative networks is an asymptotic approximation of the logarithm of the Bayes factor. The Bayes factor is the ratio of the posterior probabilities of two competing models. Let us denote by $\Delta\text{BIC}_{1,2} = \text{BIC}_1 - \text{BIC}_2$ the difference between the BIC score of network 1 and network 2. Positive values of $\Delta\text{BIC}_{1,2}$ imply

| Data set | $n$ | Data set | $n$ | Data set | $n$ | Data set | $n$ |
|---|---|---|---|---|---|---|---|
| Audio | 100 | Retail | 135 | MSWeb | 294 | Reuters-52 | 889 |
| Jester | 100 | Pumsb-star | 163 | Book | 500 | C20NG | 910 |
| Netflix | 100 | DNA | 180 | EachMovie | 500 | BBC | 1058 |
| Accidents | 111 | Kosarek | 190 | WebKB | 839 | Ad | 1556 |

Table 1: Data sets sorted according to the number $n$ of variables.

evidence in favor of network 1. The evidence in favor of network 1 is respectively [16] {weak, positive, strong, very strong} if $\Delta\mathrm{BIC}_{1,2}$ is between {0 and 2; 2 and 6; 6 and 10 ; beyond 10}.

## 5.1 Learning from datasets

We consider 16 data sets already used in the literature of structure learning, firstly introduced in [13] and [8]. We randomly split each data set into three subsets of instances. This yields 48 data sets.

The approaches for parent set identification are compared in Table 2. For each fixed structure optimization approach, we learn the network starting from the list of parent sets computed by independence selection (IS), greedy selection (GS) and sequential selection (SQ). In turn we analyze $\Delta\mathrm{BIC}_{IS,GS}$ and $\Delta\mathrm{BIC}_{IS,SQ}$. A positive $\Delta\mathrm{BIC}$ means that independence selection yields a network with higher BIC score than the network obtained using an alternative approach for parent set identification; vice versa for negative values of $\Delta\mathrm{BIC}$. In most cases (see Table 2) $\Delta\mathrm{BIC}{>}10$, implying very strong support for the network learned using independence selection. We further analyze the results through a sign-test. The null hypothesis of the test is that the BIC score of the network learned under independence selection is smaller than or equivalent to the BIC score of the network learned using the alternative approach (greedy selection or sequential selection depending on the case). If a data set yields a $\Delta\mathrm{BIC}$ which is {very negative, strongly negative, negative, neutral}, it supports the null hypothesis. If a data sets yields a BIC score which is {positive, strongly positive, extremely positive}, it supports the alternative hypothesis. Under any fixed structure solver, the sign test *rejects* the null hypothesis, providing significant evidence in favor of independence selection. In the following when we further cite the sign test we refer to same type of analysis: the sign test analyzes the counts of the $\Delta\mathrm{BIC}$ which are in favor and against a given method.

As for structure optimization, ASOBS achieves higher BIC score than OBS in *all* the 48 data sets, under *every* chosen approach for parent set identification. These results confirm the improvement of ASOBS over OBS, theoretically proven in Section 4. In most cases the $\Delta\mathrm{BIC}$ in favor of ASOBS is larger than 10. The difference in favor of ASOBS is significant (sign test, $p < 0.01$) under every chosen approach for parent set identification.

We now compare ASOBS and Gobnilp. On the smaller data sets (27 data sets with $n < 500$), Gobnilp significantly outperforms (sign test, $p < 0.01$) ASOBS under every chosen approach for parent set identification. On most of such data sets, the $\Delta\mathrm{BIC}$ in favor of the network learned by Gobnilp is larger than 10. This outcome is expected, as Gobnilp is an exact solver and those data

| structure solver | Gobnilp | | ASOBS | | OBS | |
|---|---|---|---|---|---|---|
| parent identification: IS vs | GS | SQ | GS | SQ | GS | SQ |
| **$\Delta$BIC** (K) | | | | | | |
| Very positive (K >10) | 44 | 38 | 44 | 30 | 44 | 32 |
| Strongly positive (6<K <10) | 0 | 0 | 0 | 4 | 1 | 0 |
| Positive (2 <K <6) | 0 | 4 | 2 | 3 | 0 | 2 |
| Neutral (-2 <K <2) | 2 | 3 | 0 | 4 | 2 | 4 |
| Negative (-6 <K <-2) | 0 | 1 | 2 | 1 | 0 | 2 |
| Strongly negative (-10 <K <-6) | 1 | 1 | 0 | 5 | 0 | 4 |
| Very negative (K <-10) | 1 | 1 | 0 | 1 | 1 | 4 |
| *p*-value | **<0.01** | **<0.01** | **<0.01** | **<0.01** | **<0.01** | **<0.01** |

Table 2: Comparison of the approaches for parent set identification on 48 data sets. Given any fixed solver for structural optimization, IS results in significantly higher BIC scores than both GS and SQ.

| parent identification | Independence sel. | | Forward sel | | Sequential sel. | |
|---|---|---|---|---|---|---|
| structure solver:   AS vs | GP | OB | GP | OB | GP | OB |
| **ΔBIC** (K) | | | | | | |
| Very positive (K >10) | 21 | 21 | 20 | 21 | 19 | 21 |
| Strongly positive (6<K<10) | 0 | 0 | 0 | 0 | 0 | 0 |
| Positive (2<K<6) | 0 | 0 | 0 | 0 | 0 | 0 |
| Neutral (-2<K<2) | 0 | 0 | 0 | 0 | 0 | 0 |
| Negative (-6<K<-2) | 0 | 0 | 0 | 0 | 0 | 0 |
| Strongly negative (-10<K<-6) | 0 | 0 | 0 | 0 | 0 | 0 |
| Very negative (K<-10) | 0 | 0 | 1 | 0 | 2 | 0 |
| | | | | | | |
| *p*-value | **<0.01** | **<0.01** | **<0.01** | **<0.01** | **<0.01** | **<0.01** |

Table 3: Comparison between the structure optimization approaches on the 21 data sets with $n \geq 500$. ASOBS (AS) outperforms both Gobnilp (GB) and OBS (OB), under any chosen approach for parent set identification.

sets imply a relatively reduced search space. However the focus of this paper is on *large* data sets. On the 21 data sets with $n \geq 500$, ASOBS outperforms Gobnilp (sign test, $p < 0.01$) under every chosen approach for parent set identification (Table 3).

### 5.2   Learning from data sets sampled from known networks

In the next experiments we create data sets by sampling from known networks. We take the largest networks available in the literature: [4] *andes* (n=223), *diabetes* (n=413), *pigs* (n=441), *link* (n=724), *munin* (n=1041). Additionally we randomly generate other 15 networks: five networks of size *2000*, five networks of size *4000*, five networks of size *10000*. Each variable has a number of states randomly drawn from 2 to 4 and a number of parents randomly drawn from 0 to 6. Overall we consider 20 networks. From each network we sample a data set of 5000 instances.

We perform experiments and analysis as in the previous section. For the sake of brevity we do not add further tables of results. As for parent set identification, independence selection outperforms both greedy selection and sequential selection. The difference in favor of independence selection is significant (sign test, $p$-value <0.01) under every chosen structure optimization approach. The ΔBIC of the learned network is >10 in most cases. Take for instance Gobnilp for structure optimization. Then independence selection yields a ΔBIC>10 in 18/20 cases when compared to GS and ΔBIC>10 in 19/20 cases when compared to SQ. Similar results are obtained using the other solvers for structure optimization.

Strong results support also ASOBS against OBS and Gobnilp. Under every approach for parent set identification, ΔBIC>10 is obtained in 20/20 cases when comparing ASOBS and OBS. The number of cases in which ASOBS obtains ΔBIC>10 when compared against Gobnilp ranges between 17/20 and 19/20 depending on the approach adopted for parent set selection. The superiority of ASOBS over both OBS and Gobnilp is significant (sign test, $p < 0.01$) under every approach for parent set identification.

Moreover, we measured the Hamming distance between the moralized true structure and the learned structure. On the 21 data sets with $n \geq 500$ ASOBS outperforms Gobnilp and OBS and IS outperforms GS and SQ (sign test, $p < 0.01$). The novel framework is thus superior in terms of both score and correctness of the retrieved structure.

## 6   Conclusion and future work

Our novel approximated approach for structural learning of Bayesian Networks scales up to thousands of nodes without constraints on the maximum in-degree. The current results refer to the BIC score, but in future the methodology could be extended to other scoring functions.

**Acknowledgments**

Work partially supported by the Swiss NSF grant n. 200021_146606 / 1.

## Footnotes

*Istituto Dalle Molle di studi sull'Intelligenza Artificiale (IDSIA)

†Scuola universitaria professionale della Svizzera italiana (SUPSI)

‡Università della Svizzera italiana (USI)

[1]`http://www.cs.york.ac.uk/aig/sw/gobnilp/`

[2]`http://blip.idsia.ch`

[3]$O(\cdot), \Omega(\cdot)$ and $\Theta(\cdot)$ shall be understood as usual asymptotic notation functions.

[4] `http://www.bnlearn.com/bnrepository/`

# References

[1] M. Bartlett and J. Cussens. Integer linear programming for the Bayesian network structure learning problem. *Artificial Intelligence*, 2015. in press.

[2] D. M. Chickering, C. Meek, and D. Heckerman. Large-sample learning of Bayesian networks is hard. In *Proceedings of the 19st Conference on Uncertainty in Artificial Intelligence*, UAI-03, pages 124–133. Morgan Kaufmann, 2003.

[3] G. F. Cooper and E. Herskovits. A Bayesian method for the induction of probabilistic networks from data. *Machine Learning*, 9(4):309–347, 1992.

[4] J. Cussens. Bayesian network learning with cutting planes. In *Proceedings of the 27st Conference Annual Conference on Uncertainty in Artificial Intelligence*, UAI-11, pages 153–160. AUAI Press, 2011.

[5] J. Cussens, B. Malone, and C. Yuan. IJCAI 2013 tutorial on optimal algorithms for learning Bayesian networks (https://sites.google.com/site/ijcai2013bns/slides), 2013.

[6] C. P. de Campos and Q. Ji. Efficient structure learning of Bayesian networks using constraints. *Journal of Machine Learning Research*, 12:663–689, 2011.

[7] C. P. de Campos, Z. Zeng, and Q. Ji. Structure learning of Bayesian networks using constraints. In *Proceedings of the 26st Annual International Conference on Machine Learning*, ICML-09, pages 113–120, 2009.

[8] J. V. Haaren and J. Davis. Markov network structure learning: A randomized feature generation approach. In *Proceedings of the 26st AAAI Conference on Artificial Intelligence*, 2012.

[9] D. Heckerman, D. Geiger, and D.M. Chickering. Learning Bayesian networks: The combination of knowledge and statistical data. *Machine Learning*, 20:197–243, 1995.

[10] T. Jaakkola, D. Sontag, A. Globerson, and M. Meila. Learning Bayesian Network Structure using LP Relaxations. In *Proceedings of the 13st International Conference on Artificial Intelligence and Statistics*, AISTATS-10, pages 358–365, 2010.

[11] M. Koivisto. Parent assignment is hard for the MDL, AIC, and NML costs. In *Proceedings of the 19st annual conference on Learning Theory*, pages 289–303. Springer-Verlag, 2006.

[12] M. Koivisto and K. Sood. Exact Bayesian Structure Discovery in Bayesian Networks. *Journal of Machine Learning Research*, 5:549–573, 2004.

[13] D. Lowd and J. Davis. Learning Markov network structure with decision trees. In Geoffrey I. Webb, Bing Liu 0001, Chengqi Zhang, Dimitrios Gunopulos, and Xindong Wu, editors, *Proceedings of the 10st Int. Conference on Data Mining (ICDM2010)*, pages 334–343, 2010.

[14] W. J. McGill. Multivariate information transmission. *Psychometrika*, 19(2):97–116, 1954.

[15] A. Moore and W. Wong. Optimal reinsertion: A new search operator for accelerated and more accurate Bayesian network structure learning. In T. Fawcett and N. Mishra, editors, *Proceedings of the 20st International Conference on Machine Learning*, ICML-03, pages 552–559, Menlo Park, California, August 2003. AAAI Press.

[16] A. E. Raftery. Bayesian model selection in social research. *Sociological methodology*, 25:111–164, 1995.

[17] T. Silander and P. Myllymaki. A simple approach for finding the globally optimal Bayesian network structure. In *Proceedings of the 22nd Conference on Uncertainty in Artificial Intelligence*, UAI-06, pages 445–452, 2006.

[18] M. Teyssier and D. Koller. Ordering-based search: A simple and effective algorithm for learning Bayesian networks. In *Proceedings of the 21st Conference on Uncertainty in Artificial Intelligence*, UAI-05, pages 584–590, 2005.

[19] C. Yuan and B. Malone. An improved admissible heuristic for learning optimal Bayesian networks. In *Proceedings of the 28st Conference on Uncertainty in Artificial Intelligence*, UAI-12, 2012.

[20] C. Yuan and B. Malone. Learning optimal Bayesian networks: A shortest path perspective. *Journal of Artificial Intelligence Research*, 48:23–65, 2013.

